# Basis-Function Trees as a Generalization of Local Variable Selection Methods for Function Approximation

Terence D. Sanger
Dept. Electrical Engineering and Computer Science
Massachusetts Institute of Technology, E25-534
Cambridge, MA 02139

## Abstract

Local variable selection has proven to be a powerful technique for approximating functions in high-dimensional spaces. It is used in several statistical methods, including CART, ID3, C4, MARS, and others (see the bibliography for references to these algorithms). In this paper I present a tree-structured network which is a generalization of these techniques. The network provides a framework for understanding the behavior of such algorithms and for modifying them to suit particular applications.

## 1  INTRODUCTION

Function approximation on high-dimensional spaces is often thwarted by a lack of sufficient data to adequately "fill" the space, or lack of sufficient computational resources. The technique of local variable selection provides a partial solution to these problems by attempting to approximate functions locally using fewer than the complete set of input dimensions.

Several algorithms currently exist which take advantage of local variable selection, including AID (Morgan and Sonquist, 1963, Sonquist et al., 1971), k-d Trees (Bentley, 1975), ID3 (Quinlan, 1983, Schlimmer and Fisher, 1986, Sun et al., 1988), CART (Breiman et al., 1984), C4 (Quinlan, 1987), and MARS (Friedman, 1988), as well as closely related algorithms such as GMDH (Ivakhnenko, 1971, Ikeda et al., 1976, Barron et al., 1984) and SONN (Tenorio and Lee, 1989). Most of these algorithms use tree structures to represent the sequential incorporation of increasing numbers of input variables. The differences between these techniques lie in the representation ability of the networks they generate, and the methods used to grow and prune the trees. In the following I will show why trees are a natural structure

for these techniques, and how all these algorithms can be seen as special cases of a general method I call "Basis Function Trees". I will also propose a new algorithm called an "LMS tree" which has a simple and fast network implementation.

## 2 SEPARABLE BASIS FUNCTIONS

Consider approximating a scalar function $f(x)$ of $d$-dimensional input $x$ by

$$f(x_1, \ldots, x_d) \approx \sum_{i=1}^{L} c_i \sigma_i(x_1, \ldots, x_d) \tag{1}$$

where the $\sigma_i$'s are a finite set of nonlinear basis functions, and the $c_i$'s are constant coefficients. If the $\sigma_i$'s are separable functions we can assume without loss of generality that there exists a finite set of scalar-input functions $\{\phi_n\}_{n=1}^{N}$ (which includes the constant function), such that we can write

$$\sigma_i(x_1, \ldots, x_d) = \phi_{r_1^i}(x_1) \cdots \phi_{r_d^i}(x_d) \tag{2}$$

where $x_p$ is the $p$'th component of $x$, $\phi_{r_p^i}(x_p)$ is a scalar function of scalar input $x_p$, and $r_p^i$ is an integer from 1 to $N$ specifying which function $\phi$ is chosen for the $p$'th dimension of the $i$'th basis function $\sigma_i$.

If there are $d$ input dimensions and $N$ possible scalar functions $\phi_n$, then there are $N^d$ possible basis functions $\sigma_i$. If $d$ is large, then there will be a prohibitively large number of basis functions and coefficients to compute. This is one form of Bellman's "curse of dimensionality" (Bellman, 1961). The purpose of local variable selection methods is to find a small basis which uses products of fewer than $d$ of the $\phi_n$'s. If the $\phi_n$'s are local functions, then this will select different subsets of the input variables for different ranges of their values. Most of these methods work by incrementally increasing both the number and order of the separable basis functions until the approximation error is below some threshold.

## 3 TREE STRUCTURES

Polynomials have a natural representation as a tree structure. In this representation, the output of a subtree of a node determines the weight from that node to its parent. For example, in figure 1, the subtree computes its output by summing the weights $a$ and $b$ multiplied by the inputs $x$ and $y$, and the result $ax + by$ becomes the weight from the input $x$ at the first layer. The depth of the tree gives the order of the polynomial, and a leaf at a particular depth $p$ represents a monomial of order $p$ which can be found by taking products of all inputs on the path back to the root.

Now, if we expand equation 1 to get

$$f(x_1, \ldots, x_d) \approx \sum_{i=1}^{L} c_i \phi_{r_1^i}(x_1) \cdots \phi_{r_d^i}(x_d) \tag{3}$$

we see that the approximation is a polynomial in the terms $\phi_{r_p^i}(x_p)$. So the approx-

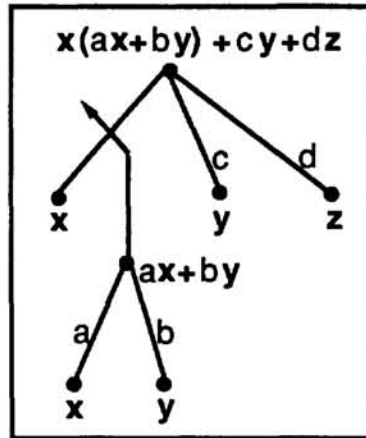

Figure 1: Tree representation of the polynomial $ax^2 + bxy + cy + dz$.

imation on separable basis functions can be described as a tree where the "inputs" are the one-dimensional functions $\phi_n(x_p)$, as in figure 2.

Most local variable selection techniques can be described in this manner. The differences in representation abilities of the different networks are determined by the choice of the one-dimensional basis functions $\phi_n$. Classification algorithms such as CART, AID, C4, or ID3 use step-functions so that the resulting approximation is piecewise constant. MARS uses a cubic spline basis so that the result is piecewise cubic.

I propose that these algorithms can be extended by considering many alternate bases. For example, for bandlimited functions the Fourier basis may be useful, for which $\phi_n(x_p) = \sin(nx_p)$ for $n$ odd, and $\cos(nx_p)$ for $n$ even. Alternatively, local Gaussians may be used to approximate a radial basis function representation. Or the bits of a binary input could be used to perform Boolean operations. I call the class of all such algorithms "Basis Function Trees" to emphasize the idea that the basis functions are arbitrary.

It is important to realize that Basis Function Trees are fundamentally different from the usual structure of multi-layer neural networks, in which the result of a computation at one layer provides the data input to the next layer. In these tree algorithms, the result of a computation at one layer determines the *weights* at the next layer. Lower levels control the behavior of the processing at higher levels, but the input data never traverses more than a single level.

## 4    WEIGHT LEARNING AND TREE GROWING

In addition to the choice of basis functions, one also has a choice of learning algorithm. Learning determines both the tree structure and the weights.

There are many ways to adjust the weights. Since the entire network is equivalent to a single-layer network described by (1), The mean-squared output error can be minimized either directly using pseudo-inverse techniques, or iteratively using

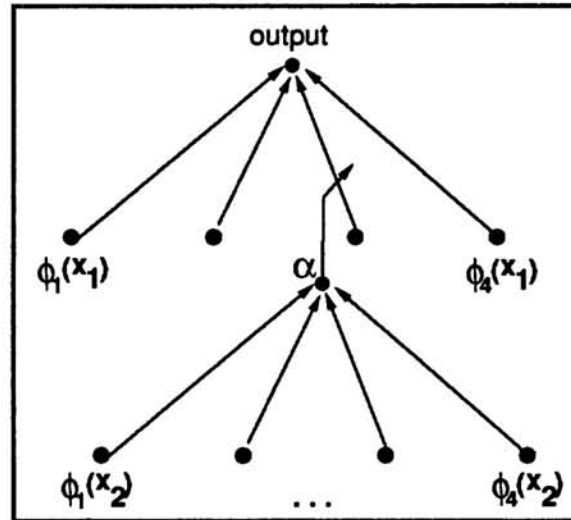

Figure 2: Tree representation of an approximation over separable basis functions.

recursive least squares (Ljung and Soderstrom, 1983) or the Widrow-Hoff LMS algorithm (Widrow and Hoff, 1960). Iterative techniques are often less robust and can take longer to converge than direct techniques, but they do not require storage of the entire data set and can adapt to nonstationary input distributions.

Since the efficiency of local variable selection methods will depend on the size of the tree, good tree growing and pruning algorithms are essential for performance. Tree-growing algorithms are often called "splitting rules", and the choice of rule should depend on the data set as well as the type of basis functions. AID and the "Regression Tree" method in CART split below the leaf with maximum mean-squared prediction error. MARS tests all possible splits by forming the new trees and estimating a "generalized cross-validation" criterion which penalizes both for output error and for increasing tree size. This method is likely to be more noise-tolerant, but it may also be significantly slower since the weights must be re-trained for every subtree which is tested. Most methods include a tree-pruning stage which attempts to reduce the size of the final tree.

## 5    LMS TREES

I now propose a new member of the class of local variable selection algorithms which I call an "LMS Tree" (Sanger, 1991, Sanger, 1990a, Sanger, 1990b). LMS Trees can use arbitrary basis functions, but they are characterized by the use of a recursive algorithm to learn the weights as well as to grow new subtrees.

The LMS tree will be built using one dimension of the input at a time. The approximation to $f(x_1, \ldots, x_d)$ using only the first dimension of the input is given by

$$f(x_1, \ldots, x_d) \approx \hat{f}(x_1) = \sum_{n=1}^{N} \alpha_n \phi_n(x_1). \tag{4}$$

I use the Widrow-Hoff LMS learning rule (Widrow and Hoff, 1960) to minimize the mean-squared approximation error based on only the first dimension:

$$\Delta \alpha_n = \eta(f(x_1, \ldots, x_d) - \hat{f}(x_1))\phi_n(x_1) \tag{5}$$

where $\eta$ is a rate term, and $\Delta \alpha_n$ is the change in the weight $\alpha_n$ made in response to the current value of $x_1$. After convergence, $\hat{f}(x_1)$ is the best approximation to $f$ based on linear combinations of $\phi_1(x_1), \ldots, \phi_N(x_1)$, and the expected value of the weight change $E[\Delta \alpha_n]$ will be zero. However, there may still be considerable variance of the weight changes, so that $E[(\Delta \alpha_n)^2] \neq 0$. The weight change variance indicates that there is "pressure" to increase or decrease the weights for certain input values, and it is related to the output error by

$$\frac{\sum_{n=1}^{N} E[(\Delta \alpha_n)^2]}{\min_{x_1} \sum_{n=1}^{N} \phi_n^2(x_1)} \geq E[(f - \hat{f})^2] \geq \max_n \frac{E[(\Delta \alpha_n)^2]}{E[(\phi_n(x_1))^2]} \tag{6}$$

(Sanger, 1990b). So the output error will be zero if and only if $E[(\Delta \alpha_n)^2] = 0$ for all $n$.

We can decrease the weight change variance by using another network based on $x_2$ to add a variable term to the weight $\alpha_{r_1}$ with largest variance, so that the new network is given by

$$\hat{f}(x_1, x_2) = \sum_{n \neq r_1} \alpha_n \phi_n(x_1) + \left(\alpha_{r_1} + \sum_{m=1}^{N} \alpha_{r_1,m}\phi_m(x_2)\right) \phi_{r_1}(x_1). \tag{7}$$

$\Delta \alpha_{r_1}$ becomes the error term used to train the second-level weights $\alpha_{r_1,m}$, so that $\Delta \alpha_{r_1,m} = \Delta \alpha_{r_1}\phi_m(x_2)$. In general, the weight change at any layer in the tree is the error term for the layer below, so that

$$\Delta \alpha_{r_1,\ldots,r_{p+1}} = \Delta \alpha_{r_1,\ldots,r_p}\phi_{r_{p+1}}(x_{p+1}) \tag{8}$$

where the root of the recursion is $\Delta \alpha_\emptyset = \eta(f(x_1, \ldots, x_d) - \hat{f})$, and $\alpha_\emptyset$ is a constant term associated with the root of the tree.

As described so far, the algorithm imposes an arbitrary ordering on the dimensions $x_1, \ldots, x_d$. This can be avoided by using all dimensions at once. The first layer tree would be formed by the additive approximation

$$f(x_1, \ldots, x_d) \approx \sum_{p=1}^{d} \sum_{n=1}^{N} \alpha_{(n,p)}\phi_n(x_p). \tag{9}$$

New subtrees would include all dimensions and could be grown below any $\phi_n(x_p)$. Since this technique generates larger trees, tree pruning becomes very important. In practice, most of the weights in large trees are often close to zero, so after a network has been trained, weights below a threshold level can be set to zero and any leaf with a zero weight can be removed.

LMS trees have the advantage of being extremely fast and easy to program. (For example, a 49-input network was trained to a size of 20 subtrees on 40,000 data

| Method | Basis Functions | Tree Growing |
|---|---|---|
| **MARS** | Truncated Cubic Polynomials | Exhaustive search for split which minimizes a cross-validation criterion |
| **CART** (Regression), **AID** | Step functions | Split leaf with largest mean-squared prediction error (= weight variance) |
| **CART** (Classification), **ID3, C4** | Step functions | Choose split which maximizes an information criterion |
| **k-d Trees** | Step functions | Split leaf with the most data points |
| **GMDH, SONN** | Data Dimensions | Find product of existing terms which maximizes correlation to desired function |
| **LMS Trees** | Any. All dimensions present at each level. | Split leaf with largest weight change variance |

Figure 3: Existing tree algorithms.

samples in approximately 30 minutes of elapsed time on a sun-4 computer. The LMS tree algorithm required 22 lines of C code (Sanger, 1990b).) The LMS rule trains the weights and automatically provides the weight change variance which is used to grow new subtrees. The data set does not have to be stored, so no memory is required at nodes. Because the weight learning and tree growing both use the recursive LMS rule, trees can adapt to slowly-varying nonstationary environments.

## 6 CONCLUSION

Figure 3 shows how several of the existing tree algorithms fit into the framework presented here. Some aspects of these algorithms are not well described by this framework. For instance, in MARS the location of the spline functions can depend on the data, so the $\phi_n$'s do not form a fixed finite basis set. GMDH is not well described by a tree structure, since new leaves can be formed by taking products of existing leaves, and thus the approximation order can increase by more than 1 as each layer is added. However, it seems that the essential features of these algorithms and the way in which they can help avoid the "curse of dimensionality" are well explained by this formulation.

### Acknowledgements

Thanks are due to John Moody for introducing me to MARS, to Chris Atkeson for introducing me to the other statistical methods, and to the many people at NIPS who gave useful comments and suggestions. The LMS Tree technique was inspired by a course at MIT taught by Chris Atkeson, Michael Jordan, and Marc Raibert. This report describes research done within the laboratory of Dr. Emilio Bizzi in the department of Brain and Cognitive Sciences at MIT. The author was supported by an NDSEG fellowship from the U.S. Air Force.

# References

Barron R. L., Mucciardi A. N., Cook F. J., Craig J. N., Barron A. R., 1984, Adaptive learning networks: Development and application in the United States of algorithms related to GMDH, In Farlow S. J., ed., *Self-Organizing Methods in Modeling*, Marcel Dekker, New York.

Bellman R. E., 1961, *Adaptive Control Processes*, Princeton Univ. Press, Princeton, NJ.

Bentley J. H., 1975, Multidimensional binary search trees used for associated searching, *Communications ACM*, 18(9):509–517.

Breiman L., Friedman J., Olshen R., Stone C. J., 1984, *Classification and Regression Trees*, Wadsworth Belmont, California.

Friedman J. H., 1988, Multivariate adaptive regression splines, Technical Report 102, Stanford Univ. Lab for Computational Statistics.

Ikeda S., Ochiai M., Sawaragi Y., 1976, Sequential GMDH algorithm and its application to river flow prediction, *IEEE Trans. Systems, Man, and Cybernetics*, SMC-6(7):473–479.

Ivakhnenko A. G., 1971, Polynomial theory of complex systems, *IEEE Trans. Systems, Man, and Cybernetics*, SMC-1(4):364–378.

Ljung L., Soderstrom T., 1983, *Theory and Practice of Recursive Identification*, MIT Press, Cambridge, MA.

Morgan J. N., Sonquist J. A., 1963, Problems in the analysis of survey data, and a proposal, *J. Am. Statistical Assoc.*, 58:415–434.

Quinlan J. R., 1983, Learning efficient classification procedures and their application to chess end games, In Michalski R. S., Carbonell J. G., Mitchell T. M., ed.s, *Machine Learning: An Artificial Intelligence Approach*, chapter 15, pages 463–482, Tioga P., Palo Alto.

Quinlan J. R., 1987, Simplifying decision trees, *Int. J. Man-Machine Studies*, 27:221–234.

Sanger T. D., 1990a, Basis-function trees for approximation in high-dimensional spaces, In Touretzky D., Elman J., Sejnowski T., Hinton G., ed.s, *Proceedings of the 1990 Connectionist Models Summer School*, pages 145–151, Morgan Kaufmann, San Mateo, CA.

Sanger T. D., 1990b, A tree-structured algorithm for function approximation in high dimensional spaces, *IEEE Trans. Neural Networks*, in press.

Sanger T. D., 1991, A tree-structured algorithm for reducing computation in networks with separable basis functions, *Neural Computation*, 3(1), in press.

Schlimmer J. C., Fisher D., 1986, A case study of incremental concept induction, In *Proc. AAAI-86, Fifth National Conference on AI*, pages 496–501, Los Altos, Morgan Kaufmann.

Sonquist J. A., Baker E. L., Morgan J. N., 1971, Searching for structure, Institute for Social Research, Univ. Michigan, Ann Arbor.

Sun G. Z., Lee Y. C., Chen H. H., 1988, A novel net that learns sequential decision process, In Anderson D. Z., ed., *Neural Information Processing Systems*, pages 760–766, American Institute of Physics, New York.

Tenorio M. F., Lee W.-T., 1989, Self organizing neural network for optimum supervised learning, Technical Report TR-EE 89-30, Purdue Univ. School of Elec. Eng.

Widrow B., Hoff M. E., 1960, Adaptive switching circuits, In *IRE WESCON Conv. Record, Part 4*, pages 96–104.
